# USING NEURAL NETWORKS TO IMPROVE COCHLEAR IMPLANT SPEECH PERCEPTION

Manoel F. Tenorio
School of Electrical Engineering
Purdue University
West Lafayette, IN 47907

## ABSTRACT

An increasing number of profoundly deaf patients suffering from sensorineural deafness are using cochlear implants as prostheses. After the implant, sound can be detected through the electrical stimulation of the remaining peripheral auditory nervous system. Although great progress has been achieved in this area, no useful speech recognition has been attained with either single or multiple channel cochlear implants.

Coding evidence suggests that it is necessary for any implant which would effectively couple with the natural speech perception system to simulate the temporal dispersion and other phenomena found in the natural receptors, and currently not implemented in any cochlear implants. To this end, it is presented here a computational model using artificial neural networks (ANN) to incorporate the natural phenomena in the artificial cochlear.

The ANN model presents a series of advantages to the implementation of such systems. First, the hardware requirements, with constraints on power, size, and processing speeds, can be taken into account together with the development of the underlining software, before the actual neural structures are totally defined. Second, the ANN model, since it is an abstraction of natural neurons, carries the necessary ingredients and is a close mapping for implementing the necessary functions. Third, some of the processing, like sorting and majority functions, could be implemented more efficiently, requiring only local decisions. Fourth, the ANN model allows function modifications through parametric modification (no software recoding), which permits a variety of fine-tuning experiments, with the opinion of the patients, to be conceived. Some of those will permit the user some freedom in system modification at real-time, allowing finer and more subjective adjustments to fit differences on the condition and operation of individual's remaining peripheral auditory system.

## 1. INTRODUCTION

The study of the model of sensory receptors can be carried out either via trying to understand how the natural receptors process incoming signals and build a representation code, or via the construction of artificial replacements. In the second case, we are interested in to what extent those artificial counterparts have the ability to replace the natural receptors.

Several groups are now carrying out the design of artificial sensors. Artificial cochleas seem to have a number of different designs and a tradition of experiments. These make them now available for widespread use as prostheses for patients who have sensorineural deafness caused by hair cell damage.

Although surgery is required for such implants, their performance has reached a level of maturity to induce patients to seek out these devices voluntarily. Unfortunately, only partial acoustic information is obtained by severely deaf patients with cochlear prosthesis. Useful patterns for speech communication are not yet fully recognizable through auditory prostheses. This problem with artificial receptors is true for both single implants, that stimulate large sections of the cochlea with signals that cover a large portion of the spectrum [4,5], and multi channel implants, that stimulate specific regions of the cochlea with specific portions of the auditory spectrum [3,13].

In this paper, we tackle the problem of artificial cochlear implants through the used of neurocomputing tools. The receptor model used here was developed by Gerald Wasserman of the Sensory Coding Laboratory, Department of Psychological Sciences, Purdue University [20], and the implants were performed by Richard Miyamoto of the Department of Otolaryngology, Indiana University Medical School [11].

The idea is to introduce with the cochlear implant, the computation that would be performed otherwise by the natural receptors. It would therefore be possible to experimentally manipulate the properties of the implant and measure the effect of coding variations on behavior. The model was constrained to be portable, simple to implant, fast enough computationally for on-line use, and built with a flexible paradigm, which would allow for modification of the different parts of the model, without having to reconstruct it entirely. In the next section, we review parts of the receptor model, and discuss the block diagram of the implant. Section 3 covers the limitations associated with the technique, and discusses the results obtained with a single neuron and one feedback loop. Section 4 discusses the implementations of these models using feedforward neural networks, and the computational advantages for doing so.

## 2. COCHLEAR IMPLANTS AND THE NEURON MODEL

Although patients cannot reliably recognize randomly chosen spoken words to them (when implanted with either multichannel or single channel devices), this is not to say that no information is extracted from speech. If the vocabulary is reduced to a limited set of words, patients perform significantly better than chance, at associating the word with a member of the set.

For these types of experiments, single channel implants correspond to reported performance of 14% to 20% better than chance, with 62% performance being the highest reported. For multiple channels, performances of 95% were reported. So far no one has investigated the differences in performance between the two types of implants. Since the two implants have so many differences, it is difficult to point out the cause for the better performance in the multiple channel case.

The results of such experiments are encouraging, and point to the fact that cochlea implants need only minor improvement to be able to mediate ad-lib speech perception successfully. Sensory coding studies have suggested a solution to the implant problem, by showing that the representation code generated by the sensory system is task dependent. This evidence came from comparison of intracellular recordings taken from a single receptor of intact subjects.

This coding evidence suggests that the temporal dispersion (time integration) found in natural receptors would be a necessary part of any

cochlear implant. Present cochlear implants have no dispersion at all. Figure 2 shows the block diagram for a representative cochlear implant, the House-Urban stimulator. The acoustic signal is picked up by the microphone, which sends it to an AM oscillator. This modulation step is necessary to induce an electro-magnetic coupling between the external and internal coil. The internal coil has been surgically implanted, and it is connected to a pair of wires implanted inside and outside the cochlea.

Just incorporating the temporal dispersion model to an existing device would not replicate the fact that in natural receptors, temporal dispersion appears in conjunction to other operations which are strongly non linear. There are operations like selection of a portion of the spectrum, rectification, compression, and time-dispersion to be considered.

In figure 3, a modified implant is shown, which takes into consideration some of these operations. It is depicted as a single-channel implant, although the ultimate goal is to make it multichannel. Details of the operation of this device can be found elsewhere [21]. Here, it is important to mention that the implant would also have a compression/rectification function, and it would receive a feedback from the integrator stage in order to control its gain.

## 3. CHARACTERISTICS AND RESULTS OF THE IMPLANTS

The above model has been implemented as an off-line process, and then the patients were exposed to a preprocessed signal which emulated the operation of the device. It is not easy to define the amount of feedback needed in the system or the amount of time dispersion. It could also be that these parameters are variable across different conditions. Another variance in the experiment is the amount of damage (and type) among different individuals. So, these parameters have to be determined clinically.

The coupling between the artificial receptor and the natural system also presents problems. If a physical connection is used, it increases the risk of infections. When inductive methods are used, the coupling is never ideal. If portability and limited power is of concern in the implementation, then the limited energy available for coupling has to be used very effectively.

The computation of the receptor model has to be made in a way to allow for fast implementation. The signal transformation is to be computed on-line. Also, the results from clinical studies should be able to be incorporated fairly easily without having to reengineer the implant.

Now we present the results of the implementation of the transfer function of figure 4. Patients, drawn from a population described elsewhere [11,12,14], were given spoken sentences processed off-line, and simultaneously presented with a couple of words related to the context. Only one of them was the correct answer. The patient had two buttons, one for each alternative; he/she was to press the button which corresponded to the correct alternative. The results are shown in the tables below.

Patient 1 (Average of the population)

|  | Percentage of correct alternatives |  |
|---|---|---|
| Dispersion |  |  |
| No disp. | 67% |  |
| 0.1 msec | 78% |  |
| 0.3 msec | 85% | Best performance |

| | |
|---|---|
| 1 msec | 76% |
| 3 msec | 72% |

Table I:  Phoneme discrimination in a two-alternate task.

Patient 2

| | Percentage of correct alternatives |
|---|---|
| Dispersion | |
| No disp. | 50% |
| 1.0 msec | 76%     Best performance |

Table II:  Sentence comprehension in a two-alternative task.

There were quite a lot of variations in the performance of the different patients, some been able to perform better at different dispersion and compression amounts than the average of the population. Since one cannot control the amount of damage in the system of each patient or differences in individuals, it is hard to predict the ideal values for a given patient. Nevertheless, the improvements observed are of undeniable value in improving speech perception.

## 4. THE NEUROCOMPUTING MODEL

In studying the implementation of such a system for on-line use, yet flexible enough to produce a carry-on device, we look at feedforward neurocomputer models as a possible answer. First, we wanted a model that easily produced a parallel implementation, so that the model could be expanded in a multichannel environment without compromising the speed of the system. Figure 5 shows the initial idea for the implementation of the device as a Single Instruction Multiple Data (SIMD) architecture.

The implant would be similar to the one described in Figure 4, except that the transfer function of the receptor would be performed by a two layer feed forward network (Figure 6). Since there is no way of finding out the values of compression and dispersion apart from clinical trials, or even if these values do change in certain conditions, we need to create a structure that is flexible enough to modify the program structure by simple manipulation of parameters. This is also the same problem we would face when trying to expand the system to a multichannel implant. Again, neuromorphic models provided a nice paradigm in which the dataflow and the function of the program could be altered by simple parameter (weight) change.

For this first implementation we chose to use the no-contact inductive coupling method. The drawback of this method is that all the information has to be compressed in a single channel for reliable transmission and cross talk elimination.

Since the inductive coupling of the implant is critical at every cycle, the most relevant information must be picked out of the processed signal. This information is then given all the available energy, and after all the coupling loss, it should be sufficient to provide for speech pattern discrimination. In a multichannel setting, this corresponds to doing a sorting of all the n signals in the channels, selecting the m highest signals, and adding them up for modulation. In a naive single processor implementation, this could correspond to $n^2$ comparisons, and in a multiprocessor implementation, $\log(n)$ comparisons. Both are dependent on the number of signals to be

sorted.

We needed a scheme in which the sorting time would be constant with the number of channels, and would be easily implementable in analog circuitry, in case this became a future route. Our scheme is shown in Figure 7. Each channel is connected to a threshold element, whose threshold can be varied externally. A monotonically decreasing function scans the threshold values, from the highest possible value of the output to the lowest. The output of these elements will be high corresponding to the values that are the highest first. These output are summed with a quasi-integrator with threshold set to m. This element, when high, disables the scanning functions; and it corresponds to having found the m highest signals. This sorting is independent of the number of channels.

The output of the threshold units are fed into sigma-pi units which gates the signals to be modulated. The output of these units are summed and correspond to the final processed signal (Figure 8).

The user has full control of the characteristics of this device. The number of channels can be easily altered; the number of components allowed in the modulation can be changed; the amount of gain, rectification-compression, and dispersion of each channel can also be individually controlled. The entire system is easily implementable in analog integrated circuits, once the clinical tests have determine the optimum operational characteristics.

## 5. CONCLUSION

We have shown that the study of sensory implants can enhance our understanding of the representation schemes used for natural sensory receptors. In particular, implants can be enhanced significantly if the effects of the sensory processing and transfer functions are incorporated in the model.

We have also shown that neuromorphic computing paradigm provides a parallel and easily modifiable framework for signal processing structures, with advantages that perhaps cannot be offered by other technology.

We will soon start the use of the first on-line portable model, using a single processor. This model will provide a testbed for more extensive clinical trials of the implant. We will then move to the parallel implementation, and from there, possibly move toward analog circuitry implementation.

Another route for the use of neuromorphic computing in this domain is possibly the use of sensory recordings from healthy animals to train self-organizing adaptive learning networks, in order to design the implant transfer functions.

## REFERENCES

[1]  Bilger, R.C.; Black, F.O.; Hopkinson, N.T.; and Myers, E.N., "Implanted auditory prosthesis: An evaluation of subjects presently fitted with cochlear implants," *Otolaryngology*, 1977, Vol. 84, pp. 677-682.

[2]  Bilger, R.C.; Black, F.O.; Hopkinson, N.T.; Myers, E.N.; Payne, J.L.; Stenson, N.R.; Vega, A.; and Wolf, R.V., "Evaluation of subjects presently fitted with implanted auditory prostheses," *Annals of Otology, Rhinology, and Laryngology*, 1977, Vol. 86(Supp. 38), pp. 1-176.

[3] Eddington, D.K.; Dobelle, W.H.; Brackmann, D.E.; Mladejovsky, M.G.; and Parkin, J., "Place and periodicity pitch by stimulation of multiple scala tympani electrodes in deaf volunteers," *American Society for Artificial Internal Organs, Transactions*, 1978, Vol. 24, pp. 1-5.

[4] House, W.F.; Berliner, K.; Crary, W.; Graham, M.; Luckey, R.; Norton, N.; Selters, W.; Tobin, H.; Urban, J.; and Wexler, M., "Cochlear implants," *Annals of Otology, Rhinology and Laryngology*, 1976, Vol. 85(Supp. 27), pp. 1-93.

[5] House, W.F. and Urban, J., "Long term results of electrode implantation and electronic stimulation of the cochlea in man," *Annals of Otology, Rhinology and Laryngology*, 1973, Vol. 82, No. 2, pp. 504-517.

[6] Ifukube, T. and White, R.L., "A speech processor with lateral inhibition for an eight channel cochlear implant and its evaluation," *IEEE Trans. on Biomedical Engineering*, November 1987, Vol. BME-34, No. 11.

[7] Kong, K.-L., and Wasserman, G.S., "Changing response measures alters temporal summation in the receptor and spike potentials of the *Limulus* lateral eye," *Sensory Processes*, 1978, Vol. 2, pp. 21-31. (a)

[8] Kong, K.-L., and Wasserman, G.S., "Temporal summation in the receptor potential of the *Limulus* lateral eye: Comparison between retinula and eccentric cells," *Sensory Processes*, 1978, Vol. 2, pp. 9-20. (b)

[9] Michelson, R.P., "The results of electrical stimulation of the cochlea in human sensory deafness," *Annals of Otology, Rhinology and Laryngology*, 1971, Vol. 80, pp. 914-919.

[10] Mladejovsky, M.G.; Eddington, D.K.; Dobelle, W.H.; and Brackmann, D.E., "Artificial hearing for the deaf by cochlear stimulation: Pitch modulation and some parametric thresholds," *American Society for Artificial Internal Organs, Transactions*, 1974, Vol. 21, pp. 1-7.

[11] Miyamoto, R.T.; Gossett, S.K.; Groom, G.L.; Kienle, M.L.; Pope, M.L.; and Shallop, J.K., "Cochlear implants: An auditory prosthesis for the deaf," *Journal of the Indiana State Medical Association*, 1982, Vol. 75, pp. 174-177.

[12] Miyamoto, R.T.; Myres, W.A.; Pope, M.L.; and Carotta, C.A., "Cochlear implants for deaf children," *Laryngoscope*, 1986, Vol. 96, pp. 990-996.

[13] Pialoux, P.; Chouard, C.H.; Meyer, B.; and Fugain, C., "Indications and results of the multichannel cochlear implant," *Acta Otolaryngology*, 1979, Vo.. 87, pp. 185-189.

[14] Robbins, A.M.; Osberger, M.J.; Miyamoto, R.T.; Kienle, M.J.; and Myres, W.A., "Speech-tracking performance in single-channel cochlear implant subjects," *Journal of Speech and Hearing Research*, 1985, Vol. 28, pp. 565-578.

[15] Russell, I.J. and Sellick, P.M., "The tuning properties of cochlear hair cells," in E.F. Evans and J.P. Wilson (eds.), *Psychophysics and Physiology of Hearing*, London: Academic Press, 1977.

[16] Wasserman, G.S., "*Limulus* psychophysics: Temporal summation in the ventral eye," *Journal of Experimental Psychology: General*, 1978, Vol. 107, pp. 276-286.

[17] Wasserman, G.S., "*Limulus* psychophysics: Increment threshold," *Perception & Psychophysics,* 1981, Vol. 29, pp. 251-260.

[18] Wasserman, G.S.; Felsten, G.; and Easland, G.S., "Receptor saturation and the psychophysical function," *Investigative Ophthalmology and Visual Science,* 1978, Vol. 17, p. 155 (Abstract).

[19] Wasserman, G.S.; Felsten, G.; and Easland, G.S., "The psychophysical function: Harmonizing Fechner and Stevens," *Science,* 1979, Vol. 204, pp. 85-87.

[20] Wasserman, G.S., "Cochlear implant codes and speech perception in profoundly deaf," *Bulletin of Psychonomic Society,* Vol. (18)3, 1987.

[21] Wasserman, G.S.; Wang-Bennett, L.T.; and Miyamoto, R.T., "Temporal dispersion in natural receptors and pattern discrimination mediated by artificial receptor," *Proc. of the Fechner Centennial Symposium,* Hans Buffart (Ed.), Elsevier/North Holland, Amsterdam, 1987.

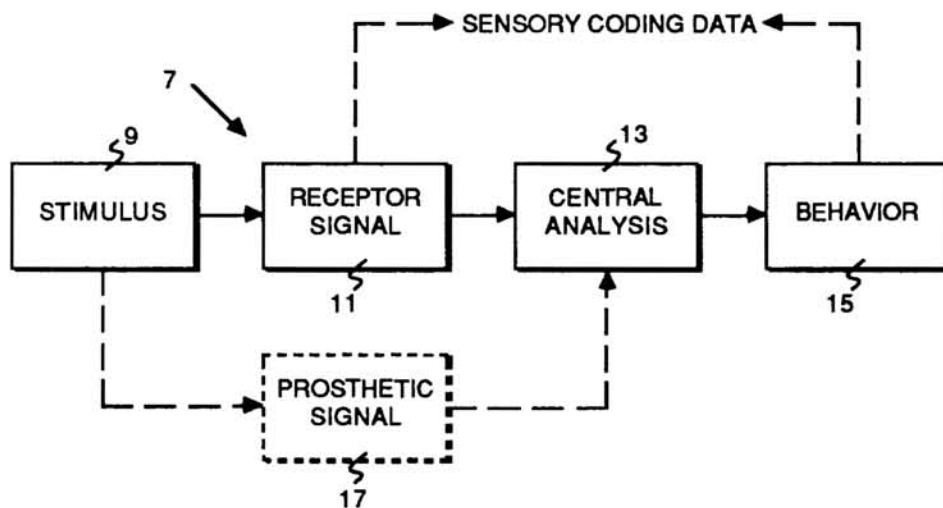

Fig. 1. Path of Natural and Prosthetic Signals.

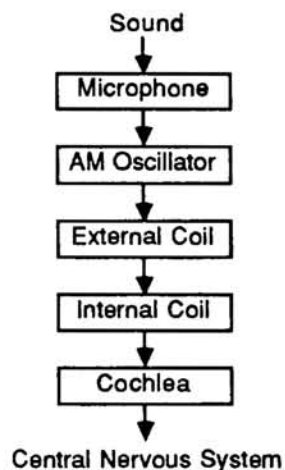

Fig. 2. The House-Urban Cochlear Implant.

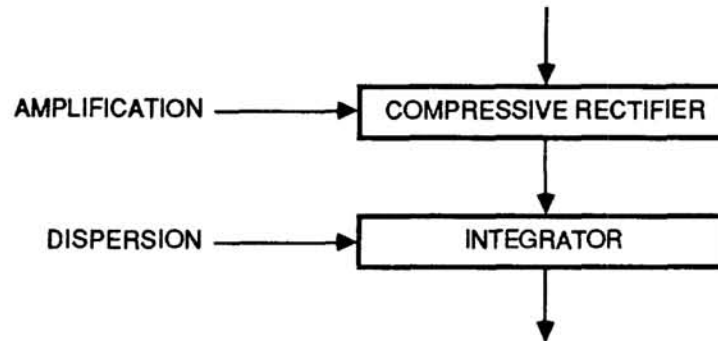

Fig. 3.  Receptor Model

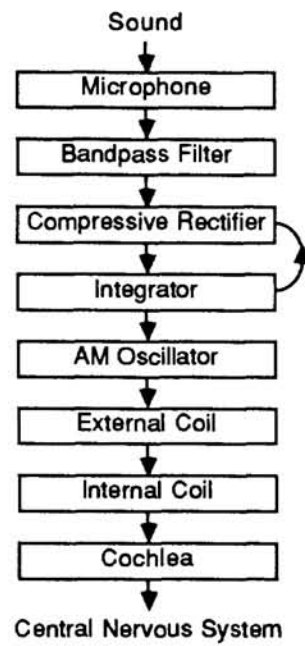

Fig. 4.  Modified Implant Model.

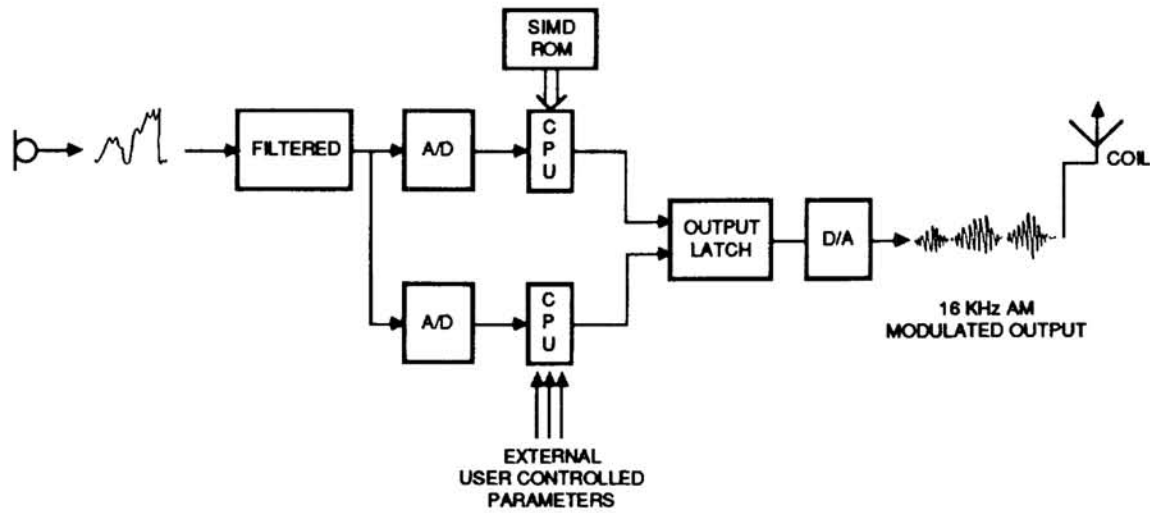

Fig. 5. Initial Concept for a SIMD Architecture.

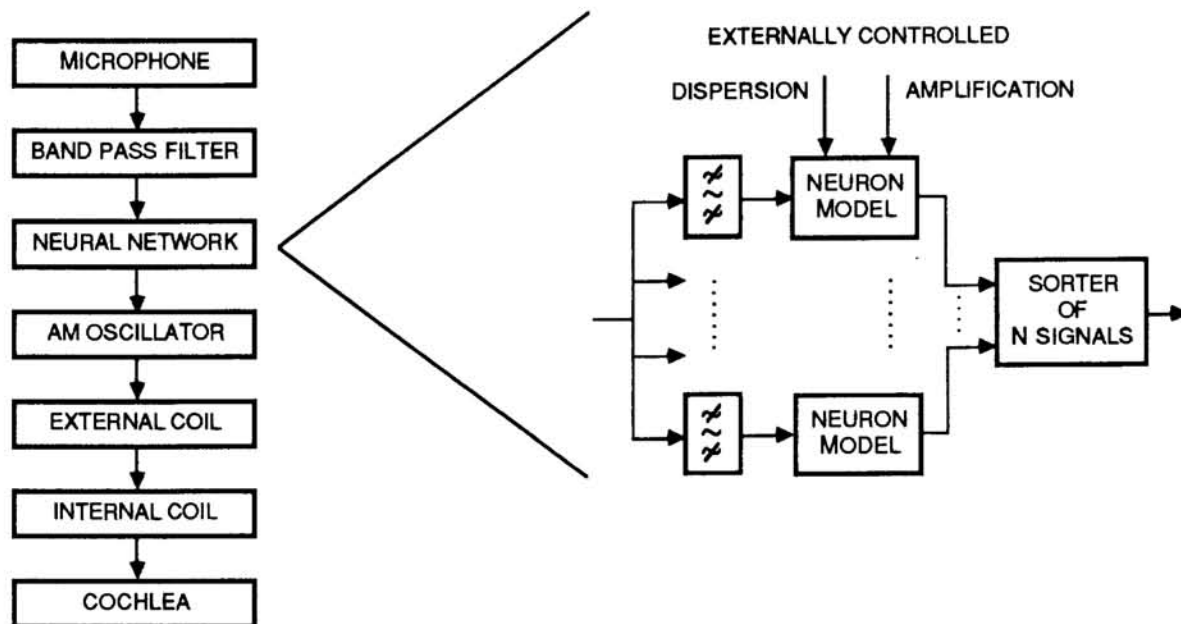

Fig. 6. Feedforward Neuron Model Implant.

## SORTER OF n SIGNALS IN O(1)

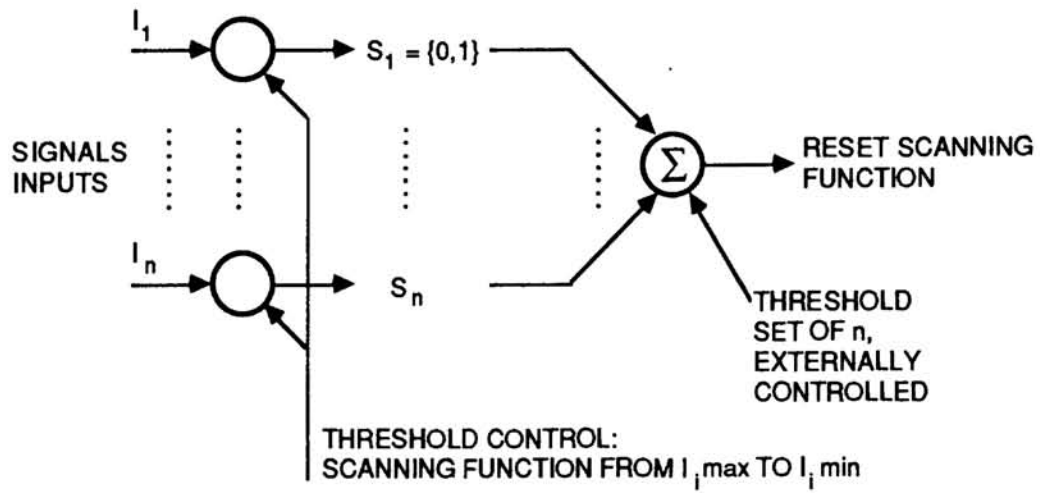

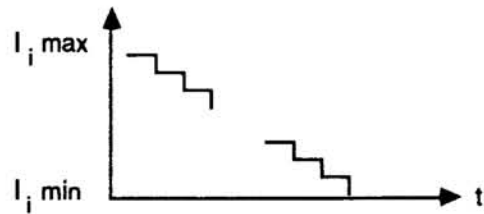

Fig. 7.  Signal Sorting Circuit.

## SIGNAL SELECTORS

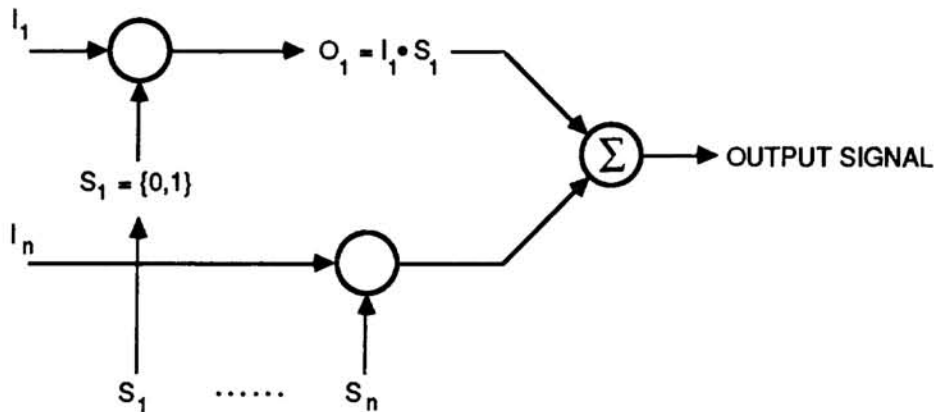

Fig. 8.  Sigma-Pi Units for Signal Composition.

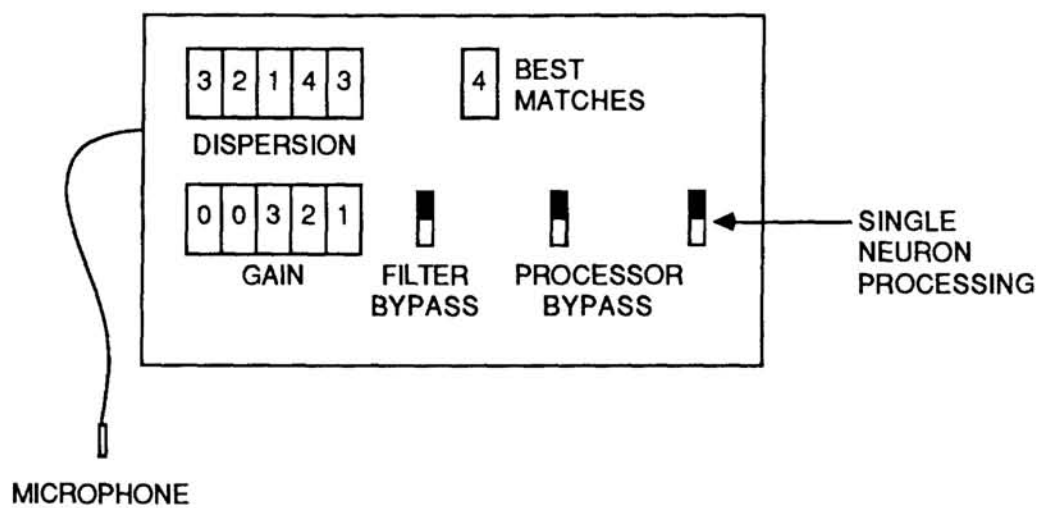

Fig. 9. Parameter Controls for Clinical Studies.